# Grammatical Inference by Attentional Control of Synchronization in an Oscillating Elman Network

**Bill Baird**
Dept Mathematics,
U.C.Berkeley,
Berkeley, Ca. 94720,
baird@math.berkeley.edu

**Todd Troyer**
Dept of Phys.,
U.C.San Francisco,
513 Parnassus Ave.
San Francisco, Ca. 94143,
todd@phy.ucsf.edu

**Frank Eeckman**
Lawrence Livermore
National Laboratory,
P.O. Box 808 (L-270),
Livermore, Ca. 94550,
eeckman@.llnl.gov

## Abstract

We show how an "Elman" network architecture, constructed from recurrently connected oscillatory associative memory network modules, can employ selective "attentional" control of synchronization to direct the flow of communication and computation within the architecture to solve a grammatical inference problem.

Previously we have shown how the discrete time "Elman" network algorithm can be implemented in a network completely described by continuous ordinary differential equations. The time steps (machine cycles) of the system are implemented by rhythmic variation (clocking) of a bifurcation parameter. In this architecture, oscillation amplitude codes the information content or activity of a module (unit), whereas phase and frequency are used to "softwire" the network. Only synchronized modules communicate by exchanging amplitude information; the activity of non-resonating modules contributes incoherent crosstalk noise.

Attentional control is modeled as a special subset of the hidden modules with ouputs which affect the resonant frequencies of other hidden modules. They control synchrony among the other modules and direct the flow of computation (attention) to effect transitions between two subgraphs of a thirteen state automaton which the system emulates to generate a Reber grammar. The internal crosstalk noise is used to drive the required random transitions of the automaton.

# 1   Introduction

Recordings of local field potentials have revealed 40 to 80 Hz oscillation in vertebrate cortex [Freeman and Baird, 1987, Gray and Singer, 1987]. The amplitude patterns of such oscillations have been shown to predict the olfactory and visual pattern recognition responses of a trained animal. There is further evidence that although the oscillatory activity appears to be roughly periodic, it is actually chaotic when examined in detail. This preliminary evidence suggests that oscillatory or chaotic network modules may form the cortical substrate for many of the sensory, motor, and cognitive functions now studied in static networks.

It remains be shown how networks with more complex dynamics can performs these operations and what possible advantages are to be gained by such complexity. We have therefore constructed a parallel distributed processing architecture that is inspired by the structure and dynamics of cerebral cortex, and applied it to the problem of grammatical inference. The construction views cortex as a set of coupled oscillatory associative memories, and is guided by the principle that attractors must be used by macroscopic systems for reliable computation in the presence of noise. This system must function reliably in the midst of noise generated by crosstalk from it's own activity. Present day digital computers are built of flip-flops which, at the level of their transistors, are continuous dissipative dynamical systems with different attractors underlying the symbols we call "0" and "1". In a similar manner, the network we have constructed is a symbol processing system, but with analog input and oscillatory subsymbolic representations.

The architecture operates as a thirteen state finite automaton that generates the symbol strings of a Reber grammar. It is designed to demonstrate and study the following issues and principles of neural computation: (1) Sequential computation with coupled associative memories. (2) Computation with attractors for reliable operation in the presence of noise. (3) Discrete time and state symbol processing arising from continuum dynamics by bifurcations of attractors. (4) Attention as selective synchronization controling communication and temporal program flow. (5) chaotic dynamics in some network modules driving randomn choice of attractors in other network modules. The first three issues have been fully addressed in a previous paper [Baird et al., 1993], and are only briefly reviewed. We focus here on the last two.

## 1.1   Attentional Processing

An important element of intra-cortical communication in the brain, and between modules in this architecture, is the ability of a module to detect and respond to the proper input signal from a particular module, when inputs from other modules irrelevant to the present computation are contributing crosstalk noise. This is smilar to the problem of coding messages in a computer architecture like the Connection Machine so that they can be picked up from the common communication buss line by the proper receiving module.

Periodic or nearly periodic (chaotic) variation of a signal introduces additional degrees of freedom that can be exploited in a computational architecture. We investigate the principle that selective control of synchronization, which we hypopthesize to be a model of "attention", can be used to solve this coding problem and control communication and program flow in an architecture with dynamic attractors.

The architecture illustrates the notion that synchronization not only "binds" sen-

sory inputs into "objects" [Gray and Singer, 1987], but binds the activity of selected cortical areas into a functional whole that directs behavior. It is a model of "attended activity" as that subset which has been included in the processing of the moment by synchronization. This is both a spatial and temporal binding. Only the inputs which are synchronized to the internal oscillatory activity of a module can effect previously learned transitions of attractors within it. For example, consider two objects in the visual field separately bound in primary visual cortex by synchronization of their components at different phases or frequencies. One object may be selectively attended to by its entrainment to oscillatory processing at higher levels such as V4 or IT. These in turn are in synchrony with oscillatory activity in motor areas to select the attractors there which are directing motor output.

In the architecture presented here, we have constrained the network dynamics so that there exist well defined notions of amplitude, phase, and frequency. The network has been designed so that amplitude codes the information content or activity of a module, whereas phase and frequency are used to "softwire" the network. An oscillatory network module has a passband outside of which it will not synchronize with an oscillatory input. Modules can therefore easily be desynchronized by perturbing their resonant frequencies. Furthermore, only synchronized modules communicate by exchanging amplitude information; the activity of non-resonating modules contributes incoherant crosstalk or noise. The flow of communication between modules can thus be controlled by controlling synchrony. By changing the intrinsic frequency of modules in a patterned way, the *effective* connectivity of the network is changed. The same hardware and connection matrix can thus subserve many different computations and patterns of interaction between modules without crosstalk problems.

The crosstalk noise is actually essential to the function of the system. It serves as the noise source for making random choices of output symbols and automaton state transitions in this architecture, as we discuss later. In cortex there is an issue as to what may constitute a source of randomness of sufficient magnitude to perturb the large ensemble behavior of neural activity at the cortical network level. It does not seem likely that the well known molecular fluctuations which are easily averaged within one or a few neurons can do the job. The architecture here models the hypothesis that deterministic chaos in the macroscopic dynamics of a network of neurons, which is the same order of magnitude as the coherant activity, can serve this purpose.

In a set of modules which is desynchronized by perturbing the resonant frequencies of the group, coherance is lost and "random" phase relations result. The character of the model time traces is irregular as seen in real neural ensemble activity. The behavior of the time traces in different modules of the architecture is similar to the temporary appearance and switching of synchronization between cortical areas seen in observations of cortical processing during sensory/motor tasks in monkeys and humans [Bressler and Nakamura, 1993]. The structure of this apparently chaotic signal and its use in network learning and operation are currently under investigation.

## 2    Normal Form Associative Memory Modules

The mathematical foundation for the construction of network modules is contained in the normal form projection algorithm [Baird and Eeckman, 1993]. This is a learning algorithm for recurrent analog neural networks which allows associative memory storage of analog patterns, continuous periodic sequences, and chaotic

attractors in the same network. An $N$ node module can be shown to function as an associative memory for up to $N/2$ oscillatory, or $N/3$ chaotic memory attractors [Baird and Eeckman, 1993]. A key feature of a net constructed by this algorithm is that the underlying dynamics is explicitly isomorphic to any of a class of standard, well understood nonlinear dynamical systems - a *normal form* [Guckenheimer and Holmes, 1983].

The network modules of this architecture were developed previously as models of olfactory cortex with distributed patterns of activity like those observed experimentally [Baird, 1990, Freeman and Baird, 1987]. Such a biological network is dynamically equivalent to a network in normal form and may easily be designed, simulated, and theoretically evaluated in these coordinates. When the intramodule competition is high, they are "memory" or winner-take-all cordinates where attractors have one oscillator at maximum amplitude, with the other amplitudes near zero. In figure two, the input and output modules are demonstrating a distributed amplitude pattern ( the symbol "T"), and the hidden and context modules are two-attractor modules in normal form coordinates showing either a right or left side active.

In this paper all networks are discussed in normal form coordinates. By analyzing the network in these coordinates, the amplitude and phase dynamics have a particularly simple interaction. When the input to a module is synchronized with its intrinsic oscillation, the amplitude of the periodic activity may be considered separately from the phase rotation. The module may then be viewed as a static network with these amplitudes as its activity.

To illustrate the behavior of individual network modules, we examine a binary (two-attractor) module; the behavior of modules with more than two attractors is similar. Such a unit is defined in polar normal form coordinates by the following equations of the Hopf normal form:

$$\dot{r}_{1i} = u_i r_{1i} - c r_{1i}^3 + (d - b sin(\omega_{clock}t))r_{1i}r_{0i}^2 + \sum_j w_{ij}^+ I_j \cos(\theta_j - \theta_{1i})$$

$$\dot{r}_{0i} = u_i r_{0i} - c r_{0i}^3 + (d - b sin(\omega_{clock}t))r_{0i}r_{1i}^2 + \sum_j w_{ij}^- I_j \cos(\theta_j - \theta_{0i})$$

$$\dot{\theta}_{1i} = \omega_i + \sum_j w_{ij}^+ (I_j/r_{1i}) \sin(\theta_j - \theta_{1i})$$

$$\dot{\theta}_{0i} = \omega_i + \sum_j w_{ij}^- (I_j/r_{0i}) \sin(\theta_j - \theta_{0i})$$

The clocked parameter $b sin(\omega_{clock}t)$ is used to implement the discrete time machine cycle of the Elman architecture as discussed later. It has lower frequency (1/10) than the intrinsic frequency of the unit $\omega_i$.

Examination of the phase equations shows that a unit has a strong tendency to synchronize with an input of similar frequency. Define the phase difference $\phi = \theta_0 - \theta_I = \theta_0 - \omega_I t$ between a unit $\theta_0$ and it's input $\theta_I$. For either side of a unit driven by an input of the same frequency, $\omega_I = \omega_0$, There is an attractor at zero phase difference $\phi = \theta_0 - \theta_I = 0$ and a repellor at $\phi = 180$ degrees. In simulations, the interconnected network of these units described below synchronizes robustly within a few cycles following a perturbation. If the frequencies of some modules of the architecture are randomly dispersed by a significant amount, $\omega_I - \omega_0 \neq 0$, phase-lags appear first, then synchronization is lost in those units. An oscillating module therefore acts as a band pass filter for oscillatory inputs.

When the oscillators are sychronized with the input, $\theta_j - \theta_{1i} = 0$, the phase terms $\cos(\theta_j - \theta_{1i}) = \cos(0) = 1$ dissappear. This leaves the amplitude equations $\dot{r}_{1i}$ and $\dot{r}_{0i}$ with static inputs $\sum_j w_{ij}^+ I_j$ and $\sum_j w_{ij}^- I_j$. Thus we have network modules which emulate static network units in their amplitude activity when fully phase-locked to their input. Amplitude information is transmitted between modules, with an oscillatory carrier.

For fixed values of the competition, in a completely synchronized system, the internal amplitude dynamics define a gradient dynamical system for a fourth order energy function. External inputs that are phase-locked to the module's intrinsic oscillation simply add a linear tilt to the landscape.

For low levels of competition, there is a broad circular valley. When tilted by external input, there is a unique equilibrium that is determined by the bias in tilt along one axis over the other. Thinking of $r_{1i}$ as the "acitivity" of the unit, this acitivity becomes a monotonically increasing function of input. The module behaves as an analog connectionist unit whose transfer function can be approximated by a sigmoid. We refer to this as the "analog" mode of operation of the module.

With high levels of competition, the unit will behave as a binary (bistable) digital flip-flop element. There are two deep potential wells, one on each axis. Hence the module performs a winner-take-all choice on the coordinates of its initial state and maintains that choice "clamped" and independent of external input. This is the "digital" or "quantized" mode of operation of a module. We think of one attractor within the unit as representing "1" (the right side in figure two) and the other as representing "0".

## 3   Elman Network of Oscillating Associative Memories

As a benchmark for the capabilities of the system, and to create a point of contact to standard network architectures, we have constructed a discrete-time recurrent "Elman" network [Elman, 1991] from oscillatory modules defined by ordinary differential equations. Previously we constructed a system which functions as the six state finite automaton that perfectly recognizes or generates the set of strings defined by the Reber grammar described in Cleeremans et. al. [Cleeremans et al., 1989]. We found the connections for this network by using the backpropagation algorithm in a static network that approximates the behavior of the amplitudes of oscillation in a fully synchronized dynamic network [Baird et al., 1993].

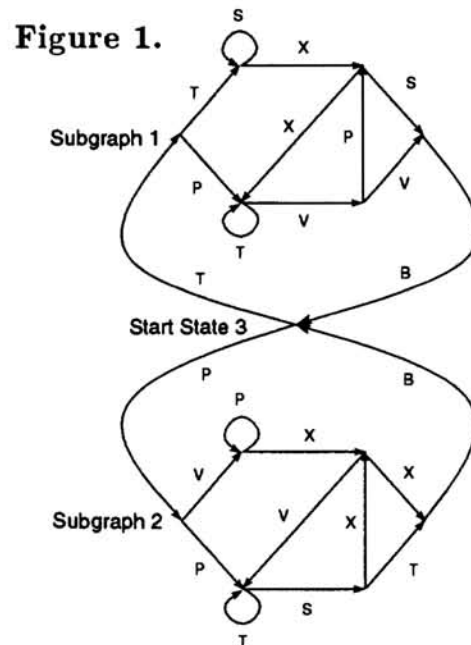

**Figure 1.**

Here we construct a system that emulates the larger 13 state automata similar (less one state) to the one studied by Cleermans, et al in the second part of their paper. The graph of this automaton consists of two subgraph branches each of which has the graph structure of the automaton learned as above, but with different assignments of transition output symbols (see fig. 1).

We use two types of modules in implementing the Elman network architecture shown in figure two below. The input and output layer each consist of a single associative memory module with six oscillatory attractors (six competing oscillatory modes), one for each of the six symbols in the grammar. The hidden and context layers consist of the binary "units" above composed of a two oscillatory attractors. The architecture consists of 14 binary modules in the hidden and context layers - three of which are special frequency control modules. The hidden and context layers are divided into four groups: the first three correspond to each of the two subgraphs plus the start state, and the fourth group consists of three special control modules, each of which has only a special control output that perturbs the resonant frequencies of the modules (by changing their values in the program) of a particular state coding group when it is at the zero attractor, as illustrated by the dotted control lines in figure two. This figure shows control unit two is at the one attractor (right side of the square active) and the hidden units coding for states of subgraph two are in synchrony with the input and output modules. Activity levels oscillate up and down through the plane of the paper. Here in midcycle, competition is high in all modules.

**Figure 2.**                    OSCILLATING ELMAN NETWORK

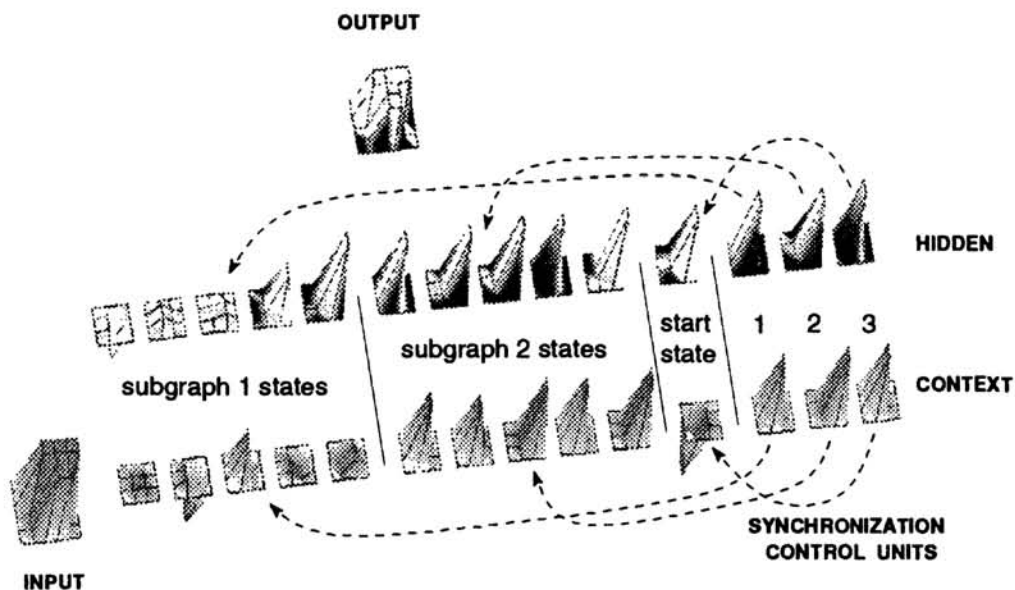

The discrete machine cycle of the Elman algorithm is implemented by the sinusoidal variation (clocking) of the bifurcation parameter in the normal form equations that determines the level of intramodule competition [Baird et al., 1993]. At the beginning of a machine cycle, when a network is generating strings, the input and context layers are at high competition and their activity is clamped at the bottom of deep basins of attraction. The hidden and output modules are at low competition and therefore behave as a traditional feedforward network free to take on analog values. In this analog mode, a real valued error can be defined for the hidden and output units and standard learning algorithms like backpropagation can be used to train the connections.

Then the situation reverses. For a Reber grammar there are always two equally possible next symbols being activated in the output layer, and we let the crosstalk noise

break this symmetry so that the winner-take-all dynamics of the output module can chose one. High competition has now also "quantized" and clamped the activity in the hidden layer to a fixed binary vector. Meanwhile, competition is lowered in the input and context layers, freeing these modules from their attractors. An identity mapping from hidden to context loads the binarized activity of the hidden layer into the context layer for the next cycle, and an additional identity mapping from the output to input module places the chosen output symbol into the input layer to begin the next cycle.

## 4   Attentional control of Synchrony

We introduce a model of attention as control of program flow by selective synchronization. The attentional controler itself is modeled in this architecture as a special set of three hidden modules with ouputs that affect the resonant frequencies of the other corresponding three subsets of hidden modules. Varying levels of intramodule competition control the large scale *direction* of information flow *between* layers of the architecture. To direct information flow on a finer scale, the attention mechanism selects a subset of modules *within* each layer whose output is effective in driving the state transition behavior of the system.

By controling the patterns of synchronization within the network we are able to generate the grammar obtained from an automaton consisting of two subgraphs connected by a single transition state (figure 1). During training we enforce a segregation of the hidden layer code for the states of the separate subgraph branches of the automaton so that different sets of synchronized modules learn to code for each subgraph of the automaton. Then the entire automaton is hand constructed with an additional hidden module for the start state between the branches. Transitions in the system from states in one subgraph of the automaton to the other are made by "attending" to the corresponding set of nodes in the hidden and context layers. This switching of the focus of attention is accomplished by changing the patterns of synchronization within the network which changes the flow of communication between modules.

Each control module modulates the intrinsic frequency of the units coding for the states a single subgraph or the unit representing the start state. The control modules respond to a particular input symbol and context to set the intrinsic frequency of the proper subset of hidden units to be equal to the input layer frequency. As described earlier, modules can easily be desynchronized by perturbing their resonant frequencies. By perturbing the frequencies of the remaining modules away from the input frequency, these modules are no longer communicating with the rest of the network. Thus coherent information flows from input to output only through one of three channels. Viewing the automata as a behavioral program, the control of synchrony constitutes a control of the program flow into its subprograms (the subgraphs of the automaton).

When either exit state of a subgraph is reached, the "B" (begin) symbol is then emitted and fed back to the input where it is connected through the first to second layer weight matrix to the attention control modules. It turns off the synchrony of the hidden states of the subgraph and allows entrainment of the start state to begin a new string of symbols. This state in turn activates both a "T" and a "P' in the output module. The symbol selected by the crosstalk noise and fed back to the input module is now connected to the control modules through the weight matrix. It desynchronizes the start state module, synchronizes in the subset of hidden units

coding for the states of the appropriate subgraph, and establishes there the start state pattern for that subgraph.

Future work will investigate the possibilities for self-organization of the patterns of synchrony and spatially segregated coding in the hidden layer during learning. The weights for entire automata, including the special attention control hidden units, should be learned at once.

## 4.1  Acknowledgments

Supported by AFOSR-91-0325, and a grant from LLNL. It is a pleasure to acknowledge the invaluable assistance of Morris Hirsch, and Walter Freeman.

# References

[Baird, 1990] Baird, B. (1990). Bifurcation and learning in network models of oscillating cortex. In Forest, S., editor, *Emergent Computation*, pages 365–384. North Holland. also in Physica D, 42.

[Baird and Eeckman, 1993] Baird, B. and Eeckman, F. H. (1993). A normal form projection algorithm for associative memory. In Hassoun, M. H., editor, *Associative Neural Memories: Theory and Implementation*, New York, NY. Oxford University Press.

[Baird et al., 1993] Baird, B., Troyer, T., and Eeckman, F. H. (1993). Synchronization and gramatical inference in an oscillating elman network. In Hanson, S., Cowan, J., and Giles, C., editors, *Advances in Neural Information Processing Systems 5*, pages 236–244. Morgan Kaufman.

[Bressler and Nakamura, 1993] Bressler, S. and Nakamura. (1993). Interarea synchronization in Macaque neocortex during a visual discrimination task. In Eeckman,F. H., and Bower, J., editors, *Computation and Neural Systems*, page 515. Kluwer.

[Cleeremans et al., 1989] Cleeremans, A., Servan-Schreiber, D., and McClelland, J. (1989). Finite state automata and simple recurrent networks. *Neural Computation*, 1(3):372–381.

[Elman, 1991] Elman, J. (1991). Distributed representations, simple recurrent networks and grammatical structure. *Machine Learning*, 7(2/3):91.

[Freeman and Baird, 1987] Freeman, W. and Baird, B. (1987). Relation of olfactory EEG to behavior: Spatial analysis. *Behavioral Neuroscience*, 101:393–408.

[Gray and Singer, 1987] Gray, C. M. and Singer, W. (1987). Stimulus dependent neuronal oscillations in the cat visual cortex area 17. *Neuroscience [Suppl]*, 22:1301P.

[Guckenheimer and Holmes, 1983] Guckenheimer, J. and Holmes, D. (1983). *Nonlinear Oscillations, Dynamical Systems, and Bifurcations of Vector Fields*. Springer, New York.